# Learning Rankings via Convex Hull Separation

**Glenn Fung, Rómer Rosales, Balaji Krishnapuram**
Computer Aided Diagnosis, Siemens Medical Solutions USA, Malvern, PA 19355
`{glenn.fung, romer.rosales, balaji.krishnapuram}@siemens.com`

## Abstract

We propose efficient algorithms for learning ranking functions from order constraints between sets—*i.e.* classes—of training samples. Our algorithms may be used for maximizing the generalized Wilcoxon Mann Whitney statistic that accounts for the partial ordering of the classes: special cases include maximizing the area under the ROC curve for binary classification and its generalization for ordinal regression. Experiments on public benchmarks indicate that: (a) the proposed algorithm is at least as accurate as the current state-of-the-art; (b) computationally, it is several orders of magnitude faster and—unlike current methods—it is easily able to handle even large datasets with over 20,000 samples.

## 1  Introduction

Many machine learning applications depend on accurately ordering the elements of a set based on the known ordering of only some of its elements. In the literature, variants of this problem have been referred to as ordinal regression, ranking, and learning of preference relations. Formally, we want to find a function $f : \Re^n \to \Re$ such that, for a set of test samples $\{x_k \in \Re^n\}$, the output of the function $f(x_k)$ can be sorted to obtain a ranking. In order to learn such a function we are provided with training data, $A$, containing $S$ sets (or classes) of training samples: $A = \bigcup_{j=1}^{S}(A^j = \{x_i^j\}_{i=1}^{m_j})$, where the $j$-th set $A^j$ contains $m_j$ samples, so that we have a total of $m = \sum_{j=1}^{S} m_j$ samples in $A$. Further, we are also provided with a directed *order graph* $G = (\mathcal{S}, \mathcal{E})$ each of whose vertices corresponds to a class $A^j$, and the existence of a directed edge $\mathcal{E}_{PQ}$—corresponding to $A^P \to A^Q$—means that all training samples $x_p \in A^P$ should be ranked higher than any sample $x_q \in A^Q$: *i.e.* $\forall (x_p \in A^P, x_q \in A^Q),\ f(x_p) \le f(x_q)$.

In general the number of constraints on the ranking function grows as $\mathcal{O}(m^2)$ so that naive solutions are computationally infeasible even for moderate sized training sets with a few thousand samples. Hence, we propose a *more stringent* problem with a larger (infinite) set of constraints, that is nevertheless much more tractably solved. In particular, we modify the constraints to: $\forall (x_p \in CH(A^P), x_q \in CH(A^Q)),\ f(x_p) \le f(x_q)$, where $CH(A^j)$ denotes the set of all points in the convex hull of $A^j$.

We show how this leads to: (a) a family of approximations to the original problem; and (b) considerably more efficient solutions that still enforce all of the original inter-group order constraints. Notice that, this formulation subsumes the standard ranking problem (*e.g.* [4]) as a special case when each set $A^j$ is reduced to a singleton and the order graph is equal to

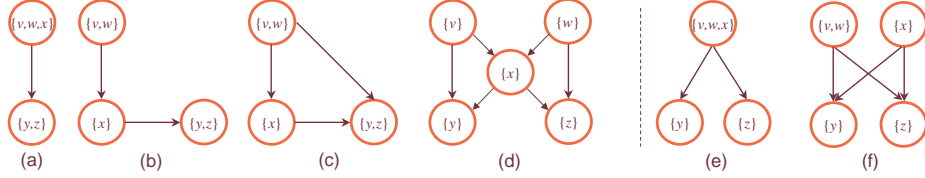

Figure 1: Various instances of the proposed ranking problem consistent with the training set $\{v, w, x, y, z\}$ satisfying $v > w > x > y > z$. Each problem instance is defined by an order graph. (a-d) A succession of order graphs with an increasing number of constraints (e-f) Two order graphs defining the same partial ordering but different problem instances.

a full graph. However, as illustrated in Figure 1, the formulation is more general and does not require a *total ordering* of the sets of training samples $A^j$, *i.e.* it allows any order graph $G$ to be incorporated into the problem.

## 1.1 Generalized Wilcoxon-Mann-Whitney Statistics

A distinction is usually made between classification and ordinal regression methods on one hand, and ranking on the other. In particular, the loss functions used for classification and ordinal regression evaluate whether each test sample is correctly classified: in other words, the loss functions that are used to evaluate these algorithms—*e.g.* the 0–1 loss for binary classification—are computed for every sample individually, and then averaged over the training or test set.

By contrast, bipartite ranking solutions are evaluated using the *Wilcoxon-Mann-Whitney* (WMW) statistic which measures the (sample averaged) probability that any *pair of samples* is ordered correctly; intuitively, the WMW statistic may be interpreted as the *area under the ROC curve* (AUC). We define a slight generalization of the WMW statistic that accounts for our notion of class-ordering:

$$WMW(f, A) = \sum_{\mathcal{E}_{ij}} \frac{\sum_{k=1}^{m_i} \sum_{l=1}^{m_j} \delta\left(f(x_k^i) < f(x_l^j)\right)}{\sum_{k=1}^{m_i} \sum_{l=1}^{m_j} 1}.$$

Hence, if a sample is individually misclassified because it falls on the wrong side of the decision boundary between classes it incurs a penalty in ordinal regression, whereas, in ranking, it may be possible that it is still correctly ordered with respect to every other test sample, and thus it may incur no penalty in the WMW statistic.

## 1.2 Previous Work

**Ordinal regression and methods for handling structured output classes**: For a classic description of generalized linear models for ordinal regression, see [11]. A non-parametric Bayesian model for ordinal regression based on Gaussian processes (GP) was defined [1]. Several recent machine learning papers consider structured output classes: *e.g.* [13] presents SVM based algorithms for handling structured and interdependent output spaces, and [5] discusses automatic document categorization into pre-defined hierarchies or taxonomies of topics.

**Learning Rankings:** The problem of learning rankings was first treated as a classification problem on pairs of objects by Herbrich [4] and subsequently used on a web page ranking task by Joachims [6]; a variety of authors have investigated this approach recently. The major advantage of this approach is that it considers a more explicit notion of ordering—However, the naive optimization strategy proposed there suffers from the $\mathcal{O}(m^2)$ growth

in the number of constraints mentioned in the previous section. This computational bur-den renders these methods impractical even for medium sized datasets with a few thousand samples. In other related work, boosting methods have been proposed for learning prefer-ences [3], and a combinatorial structure called the ranking poset was used for conditional modeling of partially ranked data[8], in the context of combining ranked sets of web pages produced by various web-page search engines. Another, less related, approach is [2].

**Relationship to the proposed work:** Our algorithm penalizes wrong ordering of pairs of training instances in order to learn ranking functions (similar to [4]), but in addition, it can also utilize the notion of a structured class order graph. Nevertheless, using a formula-tion based on constraints over convex hulls of the training classes, our method avoids the prohibitive computational complexity of the previous algorithms for ranking.

## 1.3 Notation and Background

In the following, vectors will be assumed to be column vectors unless transposed to a row vector by a prime superscript $'$. For a vector $x$ in the $n$-dimensional real space $\Re^n$, the cardinality of a set $A$ will be denoted by #(A). The scalar (inner) product of two vectors $x$ and $y$ in the $n$-dimensional real space $\Re^n$ will be denoted by $x'y$ and the 2-norm of $x$ will be denoted by $\|x\|$. For a matrix $A \in \Re^{m \times n}$, $A_i$ is the $i$th row of $A$ which is a row vector in $\Re^n$, while $A_{.j}$ is the $j$th column of $A$. A column vector of ones of arbitrary dimension will be denoted by $e$. For $A \in \Re^{m \times n}$ and $B \in \Re^{n \times k}$, the kernel $K(A, B)$ maps $\Re^{m \times n} \times \Re^{n \times k}$ into $\Re^{m \times k}$. In particular, if $x$ and $y$ are column vectors in $\Re^n$ then, $K(x', y)$ is a real number, $K(x', A')$ is a row vector in $\Re^m$ and $K(A, A')$ is an $m \times m$ matrix. The identity matrix of arbitrary dimension will be denoted by $I$.

## 2 Convex Hull formulation

We are interested in learning a ranking function $f : \Re^n \to \Re$ given known ranking rela-tionships between some *training instances* $A_i, A_j \subset A$. Let the ranking relationships be specified by a set $\mathcal{E} = \{(i, j) | A_i \prec A_j\}$

To begin with, let us consider the *linearly separable* binary ranking case which is equivalent to the problem of classifying $m$ points in the $n$-dimensional real space $\Re^n$, represented by the $m \times n$ matrix $A$, according to membership of each point $x = A_i$ in the class $A^+$ or $A^-$ as specified by a given vector of labels $d$. In others words, for binary classifiers, we want a linear ranking function $f_w(x) = w'x$ that satisfies the following constraints:

$$\forall (x^+ \in A^+, x^- \in A^-), \ f(x^-) \le f(x^+) \Rightarrow f(x^-) - f(x^+) = w'x^- - w'x^+ \le -1 \le 0.$$
(1)

Clearly, the number of constraints grows as $O(m^+ m^-)$, which is roughly quadratic in the number of training samples (unless we have severe class imbalance). While easily overcome–based on additional insights–in the separable problem, in the non-separable case, the quadratic growth in the number of constraints poses huge computational burdens on the optimization algorithm; indeed direct optimization with these constraints is infeasi-ble even for moderate sized problems. We overcome this computational problem based on **three key insights** that are explained below.

**First**, notice that (by negation) the feasibility constraints in (1) can also be defined as:

$$\forall (x^+ \in A^+, x^- \in A^-), w'x^- - w'x^+ \le -1 \Leftrightarrow \nexists(x^+ \in A^+, x^- \in A^-), w'x^- - w'x^+ > -1.$$

In other words, a solution $w$ is feasible iff there exist no pair of samples from the two classes such that $f_w(.)$ orders them incorrectly.

**Second**, we will make the constraints in (1) *more stringent*: instead of requiring that equa-tion (1) be satisfied for each possible pair $(x^+ \in A^+, x^- \in A^-)$ in the training set, we will

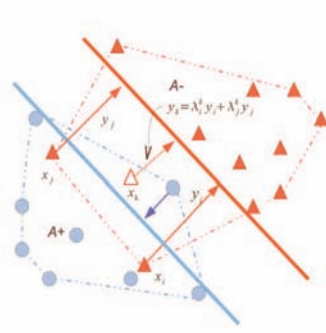

Figure 2: Example binary problem where points belonging to the $A^+$ and $A^-$ sets are represented by blue circles and red triangles respectively. Note that two elements $x_i$ and $x_j$ of the set $A^-$ are not correctly ordered and hence generate positive values of the corresponding slack variables $y_i$ and $y_j$. Note that the point $x_k$ (hollow triangle) is in the convex hull of the set $A^-$ and hence the corresponding $y_k$ error can be writen as a convex combination ($y_k = \lambda_i^k y_i + \lambda_j^k y_j$) of the two nonzero errors corresponding to points of $A^-$

require (1) to be satisfied for each pair $(x^+ \in CH(A^+), x^- \in CH(A^-))$, where $CH(A^i)$ denotes the convex hull of the set $A^i$ [12]. Thus, our constraints become:

$$\forall(\lambda^+, \lambda^-) \quad \text{such that} \quad \left\{ \begin{array}{l} 0 \leq \lambda^+ \leq 1, \sum \lambda^+ = 1 \\ 0 \leq \lambda^- \leq 1, \sum \lambda^- = 1 \end{array} \right\}, \quad w'A^{-'}\lambda^- - w'A^{+'}\lambda^+ \leq -1 \quad (2)$$

Next, notice that all the linear inequality and equality constraints on $(\lambda^+, \lambda^-)$ may be conveniently grouped together as $B\lambda \leq b$, where,

$$\lambda = \begin{bmatrix} \lambda^- \\ \lambda^+ \end{bmatrix}_{m \times 1} \quad b^+ = \begin{bmatrix} \mathbf{0}^+_{m^+ \times 1} \\ 1 \\ -1 \end{bmatrix}_{(m^+ + 2) \times 1} \quad b^- = \begin{bmatrix} \mathbf{0}^-_{m^- \times 1} \\ 1 \\ -1 \end{bmatrix}_{(m^- + 2) \times 1} \quad b = \begin{bmatrix} b^+ \\ b^- \end{bmatrix}$$

(3)

$$B^- = \begin{bmatrix} -I_{m^-} & 0 \\ e' & 0 \\ -e' & 0 \end{bmatrix}_{(m^- + 2) \times m} \quad B^+ = \begin{bmatrix} 0 & -I_{m^+} \\ 0 & e' \\ 0 & -e' \end{bmatrix}_{(m^+ + 2) \times m} \quad B = \begin{bmatrix} B^- \\ B^+ \end{bmatrix}_{(m+4) \times m}$$

(4)

Thus, our constraints on $w$ can be written as:

$$\forall \lambda \text{ s.t. } B\lambda \leq b, \ w'A^{-'}\lambda^- - w'A^{+'}\lambda^+ \leq -1 \tag{5}$$

$$\Leftrightarrow \nexists \lambda \text{ s.t. } B\lambda \leq b, \ w'A^{-'}\lambda^- - w'A^{+'}\lambda^+ > -1 \tag{6}$$

$$\Leftrightarrow \exists u \text{ s.t. } B'u - w'[A^{-'} - A^{+'}] = 0, \ b'u \leq -1, \ u \geq 0, \tag{7}$$

Where the second equivalent form of the constraints was obtained by negation (as before), and the third equivalent form results from our **third** key insight: the application of Farka's theorem of alternatives[9]. The resulting linear system of $m$ equalities and $m + 5$ inequalities in $m + n + 4$ variables can be used while minimizing any regularizer (such as $\|w\|^2$) to obtain the linear ranking function that satisfies (1); notice, however, that we avoid the $O(m^2)$ scaling in constraints.

## 2.1 The binary non-separable case

In the non-separable case, $CH(A^+) \bigcap CH(A^-) \neq \emptyset$ so the requirements have to be relaxed by introducing slack variables. To this end, we allow one slack variable $y_i \geq 0$ for each training sample $x_i$, and consider the slack for any point *inside* the convex hull $CH(A^j)$ to also be a convex combination of $y$ (see Fig. 2). For example, this implies that

if only a subset of training samples have non-zero slacks $y_i > 0$ (*i.e.* they are possibly misclassified), then the slacks of any points inside the convex hull also only depend on those $y_i$. Thus, our constraints now become:

$$\forall \lambda \ \text{s.t.} \ B\lambda \le b, \ \ w'A^{-'}\lambda^- - w'A^{+'}\lambda^+ \le -1 + (\lambda^- y^- + \lambda^+ y^+), \ \ y^+ \ge 0, \ \ y^- \ge 0. \quad (8)$$

Applying Farka's theorem of alternatives, we get:

$$(2) \Leftrightarrow \exists u \ \text{s.t.} \ B'u - \begin{bmatrix} A^-w \\ -A^+w \end{bmatrix} + \begin{bmatrix} y^- \\ y^+ \end{bmatrix} = 0, \ \ b'u \le -1, \ \ u \ge 0 \quad (9)$$

Replacing $B$ from equation (4) and defining $u' = [u^{-'} \quad u^{+'}] \ge 0$ we get the constraints:

$$\begin{aligned}
B^{+'}u^+ + A^+w + y^+ &= 0, &(10)\\
B^{-'}u^- - A^-w + y^- &= 0, &(11)\\
b^+u^+ + b^-u^- &\le -1, \ \ u \ge 0 &(12)
\end{aligned}$$

## 2.2 The general ranking problem

Now we can extend the idea presented in the previous section for any given arbitrary directed *order graph* $G = (\mathcal{S}, \mathcal{E})$, as stated in the introduction, each of whose vertices corresponds to a class $A^j$ and the existence of a directed edge $\mathcal{E}_{ij}$ means that all training samples $x_i \in A^i$ should be ranked higher than any sample $x_j \in A^j$, that is:

$$f(x^j) \le f(x^i) \Rightarrow f(x^j) - f(x^i) = w'x^j - w'x^i \le -1 \le 0 \quad (13)$$

Analogously we obtain the following set of equations that enforced the ordering between sets $A^i$ and $A^j$:

$$\begin{aligned}
B^{i'}u^{ij} + A^iw + y^i &= 0 &(14)\\
B^{j'}\hat{u}^{ij} - A^jw + y^j &= 0 &(15)\\
b^iu^{ij} + b^j\hat{u}^{ij} &\le -1 &(16)\\
u^{ij}, \hat{u}^{ij} &\ge 0 &(17)
\end{aligned}$$

It can be shown that using the definitions of $B^i, B^j, b^i, b^j$ and the fact that $u^{ij}, \hat{u}^{ij} \ge 0$, equations (14) can be rewritten in the following way:

$$\begin{aligned}
\gamma^{ij} + A^iw + y^i &\ge 0 &(18)\\
\hat{\gamma}^{ij} - A^jw + y^j &\ge 0 &(19)\\
\gamma^{ij} + \hat{\gamma}^{ij} &\le -1 &(20)\\
y^i, y^j &\ge 0 &(21)
\end{aligned}$$

where $\gamma^{ij} = b^iu^{ij}$ and $\hat{\gamma}^{ij} = b^j\hat{u}^{ij}$. Note that enforcing the constraints defined above indeed implies the desired ordering, since we have:

$$A^iw + y^i \ge -\gamma^{ij} \ge \hat{\gamma}^{ij} + 1 \ge \hat{\gamma}^{ij} \ge A^jw - y^j$$

It is also important to note the connection with Support Vector Machines (SVM) formulation [10, 14] for the binary case. If we impose the extra constraints $-\gamma^{ij} = \gamma + 1$ and $\hat{\gamma}^{ij} = \gamma - 1$, then equations (18) imply the constraints included in the standard primal SVM formulation. To obtain a more general formulation, we can "kernelize" equations (14) by making a transformation of the variable $w$ as: $w = A'v$, where v can be interpreted as an arbitrary variable in $R^m$, This transformation can be motivated by duality theory [10], then equations (14) become:

$$\begin{aligned}
\gamma^{ij} + A^iA'v + y^i &\ge 0 &(22)\\
\hat{\gamma}^{ij} - A^jA'v + y^j &\ge 0 &(23)\\
\gamma^{ij} + \hat{\gamma}^{ij} &\le -1 &(24)\\
y^i, y^j &\ge 0 &(25)
\end{aligned}$$

If we now replace the linear kernels $A^i A'$ and $A^i A'$ by more general kernels $K(A^i, A')$ and $K(A^j, A')$ we obtain a "kernelized" version of equations (14)

$$E_{ij} \equiv \left\{ \begin{array}{ll} \gamma^{ij} + K(A^i, A')v + y^i & \geq \quad 0 \\ \hat{\gamma}^{ij} - K(A^j, A')v + y^j & \geq \quad 0 \\ \gamma^{ij} + \hat{\gamma}^{ij} & \leq \quad -1 \\ y^i, y^j & \geq \quad 0 \end{array} \right\} \tag{26}$$

Given a graph $\mathcal{G} = (\mathcal{V}, \mathcal{E})$ representing the ordering of the training data and using equations (26) , we present next, a general mathematical programming formulation the ranking problem:

$$\min_{\{v, y^i, \gamma^{ij} \mid (i,j) \in \mathcal{E}\}} \quad \nu \epsilon(y) + R(v)$$
$$\text{s.t.} \quad E_{ij} \quad \forall (i, j) \in \mathcal{E} \tag{27}$$

Where $\epsilon$ is a given loss function for the slack variables $y^i$ and $R(v)$ represents a regularizer on the normal to the hyperplane $v$. For an arbitrary kernel $K(x, x')$ the number of variables of formulation (27) is $2 * m + 2\#(\mathcal{E})$ and the number of linear equations(excluding the nonnegativity constraints) is $m\#(\mathcal{E}) + \#(\mathcal{E}) = \#(\mathcal{E})(m + 1)$. for a linear kernel i.e. $K(x, x') = xx'$ the number of variables of formulation (27) becomes $m + n + 2\#(\mathcal{E})$ and the number of linear equations remains the same. When using a linear kernel and using $\epsilon(x) = R(x) = \|x\|_2^2$, the optimization problem (27) becomes a linearly constrained quadratic optimization problem for which a unique solution exists due to the convexity of the objective function:

$$\min_{\{w, y^i, \gamma^{ij} \mid (i,j) \in \mathcal{E}\}} \quad \nu \|y\|_2^2 + \frac{1}{2} w'w$$
$$\text{s.t.} \quad E_{ij} \quad \forall (i, j) \in \mathcal{E} \tag{28}$$

Unlike other SVM-like methods for ranking that need a $O(m^2)$ number of slack variables $y$ our formulation only require one slack variable for example, only $m$ slack variables are used, giving our formulation computational advantage over ranking methods. Next, we demonstrate the effectiveness of our algorithm by comparing it to two state-of-the-art algorithms.

## 3   Experimental Evaluation

We test tested our approach in a set of nine publicly available datasets [1] shown in Tab. 1 (several large datasets are not reported since only the algorithm presented in this paper was able to run them). These datasets have been frequently used as a benchmark for ordinal regression methods (*e.g.* [1]). Here we use them for evaluating ranking performance. We compare our method against SVM for ranking (*e.g.* [4, 6]) using the SVM-light package [2] and an efficient Gaussian process method (the informative vector machine) [3] [7].

These datasets were originally designed for regression, thus the continuous *target* values for each dataset were discretized into five equal size bins. We use these bins to define our ranking constraints: all the datapoints with target value falling in the same bin were grouped together. Each dataset was divided into 10% for testing and 90% for training. Thus, the input to all of the algorithms tested was, for each point in the training set: (1) a vector in $\Re^n$ (where $n$ is different for each set) and (2) a value from 1 to 5 denoting the rank of the group to which it belongs.

Performance is defined in terms of the Wilcoxon statistic. Since we do not employ information about the ranking of the elements within each group, order constraints within a group

Table 1: Benchmark Datasets

| Name | $m$ | $n$ | Name | $m$ | $n$ |
|---|---|---|---|---|---|
| 1 Abalone | 4177 | 9 | 6 Machine-CPU | 209 | 7 |
| 2 Airplane Comp. | 950 | 10 | 7 Pyrimidines | 74 | 28 |
| 3 Auto-MPG | 392 | 8 | 8 Triazines | 186 | 61 |
| 4 CA Housing | 20640 | 9 | 9 WI Breast Cancer | 194 | 33 |
| 5 Housing-Boston | 506 | 14 | | | |

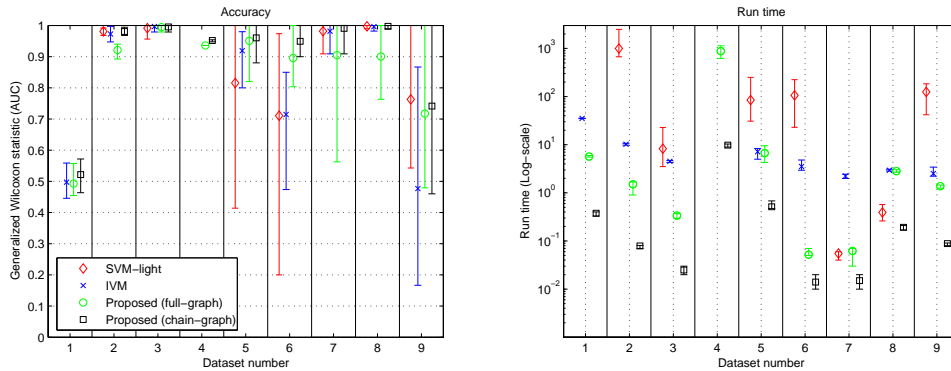

Figure 3: Experimental comparison of the ranking SVM, IVM and the proposed method on nine benchmark datasets. Along with the mean values in 10 fold cross-validation, the entire range of variation is indicated in the error-bars. (a) The overall accuracy for all the three methods is comparable. (b) The proposed method has a much lower run time than the other methods, even for the full graph case for medium to large size datasets. NOTE: Both SVM-light and IVM ran out of memory and crashed on dataset 4; on dataset 1, SVM-light failed to complete even one fold after more than 24 hours of run time, so its results could not be compiled in time for submission.

cannot be verified. Letting $b(m) = m(m-1)/2$, the total number of order constraints is equal to $b(m) - \sum_i b(m_i)$, where $m_i$ is the number of instances in group $i$.

The results for all of the algorithms are shown in Fig.3. Our formulation was tested employing two order graphs, the full directed acyclic graph and the chain graph. The performance for all datasets is generally comparable or significantly better for our algorithm (when using a chain order graph). Note that the performance for the full graph is consistently lower than that for the chain graph. Thus, interestingly enforcing more order constraints does not necessarily imply better performance. We suspect that this is due to the role that the slack variables play in both formulations, since the number of slack variables remains the same while the number of constraints increases. Adding more slack variables may positively affect performance in the full graph, but this comes at a computational cost. An interesting problem is to find the right compromise. A different but potentially related problem is that of finding *good* order graph given a dataset. Note also that the chain graph is much more stable regarding performance overall. Regarding run-time, our algorithm runs an order of magnitude faster than current implementations of state-of-the-art methods, even approximate ones (like IVM).

## 4 Discussions and future work

We propose a general method for learning a ranking function from structured order constraints on sets of training samples. The proposed algorithm was illustrated on benchmark ranking problems with two different constraint graphs: (a) a chain graph; and (b) a full

ordering graph. Although a chain graph was more accurate in the experiments shown in Figure 3, with either type of graph structure, the proposed method is at least as accurate (in terms of the WMW statistic for ordinal regression) as state-of-the-art algorithms such as the ranking-SVM and Gaussian Processes for ordinal regression.

Besides being accurate, the computational requirements of our algorithm scale much more favorably with the number of training samples as compared to other state-of-the-art methods. Indeed it was the only algorithm capable of handling several large datasets, while the other methods either crashed due to lack of memory or ran for so long that they were not practically feasible. While our experiments illustrate only specific order graphs, we stress that the method is general enough to handle arbitrary constraint relationships.

While the proposed formulation reduces the computational complexity of enforcing order constraints, it is entirely independent of the *regularizer* that is minimized (under these constraints) while learning the optimal ranking function. Though we have used a simple margin regularization (via $\|w\|^2$ in (28), and RKHS regularization in (27) in order to learn in a *supervised* setting, we can just as easily easily use a graph-Laplacian based regularizer that exploits unlabeled data, in order to learn in *semi-supervised* settings. We plan to explore this in future work.

## Footnotes

[1] Available at `http:\\www.liacc.up.pt\Ĩtorgo\Regression\DataSets.html`

[2] `http:\\www.cs.cornell.edu\People\tj\svm_light\`

[3] `http:\\www.dcs.shef.ac.uk\ neil\ivm\`

## References

[1] W. Chu and Z. Ghahramani, *Gaussian processes for ordinal regression*, Tech. report, University College London, 2004.

[2] K. Crammer and Y. Singer, *Pranking with ranking*, Neural Info. Proc. Systems, 2002.

[3] Y. Freund, R. Iyer, and R. Schapire, *An efficient boosting algorithm for combining preferences*, Journal of Machine Learning Research **4** (2003), 933–969.

[4] R. Herbrich, T. Graepel, and K. Obermayer, *Large margin rank boundaries for ordinal regression*, Advances in Large Margin Classifiers (2000), 115–132.

[5] T. Hofmann, L. Cai, and M. Ciaramita, *Learning with taxonomies: Classifying documents and words*, (NIPS) Workshop on Syntax, Semantics, and Statistics, 2003.

[6] T. Joachims, *Optimizing search engines using clickthrough data*, Proc. ACM Conference on Knowledge Discovery and Data Mining (KDD), 2002.

[7] N. Lawrence, M. Seeger, and R. Herbrich, *Fast sparse gaussian process methods: The informative vector machine*, Neural Info. Proc. Systems, 2002.

[8] G. Lebanon and J. Lafferty, *Conditional models on the ranking poset*, Neural Info. Proc. Systems, 2002.

[9] O. L. Mangasarian, *Nonlinear programming*, McGraw–Hill, New York, 1969, Reprint: SIAM Classic in Applied Mathematics 10, 1994, Philadelphia.

[10] ______ , *Generalized support vector machines*, Advances in Large Margin Classifiers, 2000, pp. 135–146.

[11] P. McCullagh and J. Nelder, *Generalized linear models*, Chapman & Hall, 1983.

[12] R. T. Rockafellar, *Convex analysis*, Princeton University Press, Princeton, New Jersey, 1970.

[13] I. Tsochantaridis, T. Hofmann, T. Joachims, and Y. Altun, *Support vector machine learning for interdependent and structured output spaces*, Int.Conf. on Machine Learning, 2004.

[14] V. N. Vapnik, *The nature of statistical learning theory*, second ed., Springer, New York, 2000.
